# Learning Dense 3D Correspondence

**Florian Steinke**[*], **Bernhard Schölkopf**[*], **Volker Blanz**[+]
[*]Max Planck Institute for Biological Cybernetics, 72076 Tübingen, Germany
`{steinke, bs}@tuebingen.mpg.de`
[+]Universität Siegen, 57068 Siegen, Germany
`blanz@mpi-sb.mpg.de`

## Abstract

Establishing correspondence between distinct objects is an important and non-trivial task: correctness of the correspondence hinges on properties which are difficult to capture in an a priori criterion. While previous work has used a priori criteria which in some cases led to very good results, the present paper explores whether it is possible to learn a combination of features that, for a given training set of aligned human heads, characterizes the notion of correct correspondence. By optimizing this criterion, we are then able to compute correspondence and morphs for novel heads.

## 1 Introduction

Establishing 3D correspondence between surfaces such as human faces is a crucial element of class-specific representations of objects in computer vision and graphics. On faces, for example, corresponding points may be the tips of the noses in 3D scans of different individuals. Dense correspondence is a mapping or "warp" from all points of a surface onto another surface (in some cases, including the present work, extending from the surface to the embedding space). Once this mapping is established, it is straightforward, for instance, to compute morphs between objects. More importantly, if correspondence mappings between a class of objects and a reference object have been established, we can represent each object by its mapping, leading to a linear representation that is able to describe also new objects of similar shape and texture (for further details, see [1]).

The practical relevance of surface correspondence has been increasing over the last years. In computer graphics, applications involve morphing, shape modeling and animation. In computer vision, an increasing number of algorithms for face and object recognition based on 2D images or 3D scans, as well as shape retrieval in databases and 3D surface reconstruction from images, rely on shape representations that are built upon dense surface correspondence.

Unlike existing algorithms that define some ad-hoc criteria for identifying corresponding points on two objects, we treat correspondence as a machine learning problem and propose a data-driven approach that learns the relevant criteria from a dataset of given object correspondences.

In stereo vision and optical flow [2, 3], a correspondence is correct if and only if it maps a point in one scene to a point in another scene which stems from the same *physical* point. In contrast, correspondence between different objects is not a well-defined problem. When two faces are compared, only some anatomically unique features such as the corners of the eyes are clearly corresponding, while it may be difficult to define how smooth regions, such as the cheeks and the forehead, are supposed to be mapped onto each other. On a more fundamental level, however, even the problem of matching the eyes is difficult to cast in a formal way, and in fact this matching involves many of the basic problems of computer vision and feature detection. In a given application, the desired correspondence can be dependent on anatomical facts, measures of shape similarity, or the overall layout of features on the surface. However, it may also depend on the properties of human perception, on functional or semantic issues, on the context within a given object class or even on social

convention. Due to the problematic and challenging nature of the correspondence problem, our correspondence learning algorithm may be a more appropriate approach than existing techniques, as it is often easier to provide a set of examples of the desired correspondences than a formal criterion for correct correspondence.

In a nutshell, the main idea of our approach is as follows. Given two objects $O_1$ and $O_2$, we are seeking a correspondence mapping $\tau$ such that certain properties of $x$ (relative to $O_1$) are preserved in $\tau(x)$ (relative to $O_2$) — they are invariant. These properties depend on the object class and as explained above, we cannot hope to characterize them comprehensively a priori. However, if we are given examples of correct and incorrect correspondences, we can attempt to *learn properties which are **invariant** for correct correspondences, while for incorrect correspondences, they are not*. We shall do this by providing a dictionary of potential properties (such as geometric features, or texture properties) and approximating a "true" property characterizing correspondence as an expansion in that dictionary. We will call this property *warp-invariant feature* and show that its computation can be cast as a problem of oriented PCA.

The remainder of the paper is structured as follows: in Section 2 we review some related work, whereas in Section 3 we set up our general framework for computing correspondence fields. Following this, we explain in Section 4 how to learn the characteristic properties for correspondence and continue to explain two new feature functions in Section 5. We give implementation details and experimental results in Section 6 and conclude in Section 7.

## 2   Related Work

The problem of establishing dense correspondence has been addressed in the domain of 2D images, on surfaces embedded in 3D space, and on volumetric data. In the image domain, correspondence from optical flow [2, 3] has been used to describe the transformations of faces with pose changes and facial expressions [4], and to describe the differences in the shapes of individual faces [5].

An algorithm for computing correspondence on parameterized 3D surfaces has been introduced for creating a class-specific representation of human faces [1] and bodies [6]. [7] propose a method that is designed to align three dimensional medical images using a mutual information criterion. Another interesting approach is [8]: they formulate the problem in a probabilistic setup and then apply standard graphical model inference algorithms to compute the correspondence. Their mesh based method uses a smoothness functional and features based on spin images. See the review [9] for an overview of a wide range of additional correspondence algorithms.

Algorithms that are applied to 3D faces typically rely on surface parameterizations, such as cylindrical coordinates, and then compute optical flow on the texture map as well as the depth image [1]. This algorithm yields plausible results, to which we will compare our method. However, the approach cannot be applied unless a parameterization is possible and the distortions are low on all elements of the object class. Even for faces this is a problem, for example around the ears, which makes a more general real 3D approach preferable. One such algorithm is presented in [10]: here, the surfaces are embedded into the surrounding space and a 3D volume deformation is computed. The use of the signed distance function as a guiding feature ensures correct surface to surface mappings. We build on this approach that is more closely presented in Section 3.

A common local geometric feature is surface curvature. Though implicit surface representations allow the extraction of such features [11], these differential geometric properties are inherently instable with respect to noise. [12] propose a related 3D geometric feature based on integrals and thus more stable to compute. We present a slightly modified version thereof which allows for a much easier computation of this feature from a signed distance function represented as a kernel expansion in comparison to a complete space voxelisation step required in [12].

## 3   General Framework For Computing Correspondence

In order to formalize our understanding of correspondence, let us assume that all the objects $O$ of class $\mathcal{O}$ are embedded in $\mathcal{X} \subseteq \mathbb{R}^3$. Given a reference object $O_r$ and a target $O_t$ the goal of computing a correspondence can then be expressed as determining the *deformation function* $\tau : \mathcal{X} \rightarrow \mathcal{X}$ which maps each point $x \in \mathcal{X}$ on $O_r$ to its corresponding point $\tau(x)$ on $O_t$.

We further assume that we can construct a *dictionary* of so-called *feature functions* $f_i : \mathcal{X} \rightarrow \mathbb{R}$, $i = 1,..,n$ capturing certain characteristic properties of the objects. [10] propose to use the signed distance function, which maps to each point $x \in \mathcal{X}$ the distance to the objects surface — with positive sign outside the shape and negative sign inside. They also use the first derivative of the signed distance function, which can be interpreted as the surface normal. In section Section 5 we will propose two additional features which are characteristic for 3D shapes, namely a curvature related feature and surface texture.

We assume that the warp-invariant feature can be represented or at least approximated by an expansion in this dictionary. Let $\gamma : \mathcal{X} \rightarrow \mathbb{R}^n$ be a weighting function describing the relative importance of the different elements of the dictionary at a given location in $\mathcal{X}$. We then express the warp-invariant feature as $f_\gamma : \mathcal{X} \rightarrow \mathbb{R}$, $f_\gamma(x) = \sum_{i=1}^n \gamma_i(x) f_i(x)$ with feature functions $f_i$ that are object specific; for the target object there is a slight modification in that the space-variant weighting $\gamma(x)$ needs to refer to the coordinates of the reference object if we want to avoid comparing apples and oranges. We thus use $f_\gamma^t(x) = \sum_{i=1}^n \gamma_i(\tau^{-1}(x)) f_i^t(x)$, where we never have to evaluate $\tau^{-1}$ since we will only require $f_\gamma^t(\tau(x))$ below.

To determine a mapping $\tau$ which will establish correct correspondences between $x$ and $\tau(x)$, we minimize the functional

$$C_{reg} \|\tau\|_{\mathcal{H}}^2 + \int_{\mathcal{X}} \left( f_\gamma^r(x) - f_\gamma^t(\tau(x)) \right)^2 d\mu(x) \tag{1}$$

The first term expresses a prior belief in a smooth deformation. This is important in regions where the objects are not sufficiently characteristic to specify a good correspondence. As we will use a Support Vector framework to represent $\tau$, smoothness can readily be expressed as the RKHS norm $\|\tau\|_{\mathcal{H}}$ of the non-parametric part of the deformation function $\tau$ (see Section 6). The second term measures the local similarity of the warp-invariant feature function extracted on the reference object $f^r$ and on the target object $f^t$ and integrates it over the volume of interest.

This formulation is a modification of [10] where two feature functions were chosen a priori (the signed distance and its derivative) and used instead of $f_\gamma$. The motivation for this is that for a correct morph, these functions should be reasonably invariant. In contrast, the present approach starts from the notion of invariance and estimates a location-dependent linear combination of feature functions with a maximal degree of invariance for correct correspondences (cf. next section). We consider location-dependent linear combinations since one cannot expect that all the feature functions that define correspondence are equally important for all points of an object. For example color may be more characteristic around the lips or the eyes than on the forehead.

This comes at the cost, however, of increasing the number of free parameters, leading to potential difficulties when performing model selection. As discussed above, it is unclear how to characterize and evaluate correspondence in a principled way. The authors of [10] propose a strategy based on a two-way morph: they first compute a deformation from the reference object to the target, and afterwards vice versa. A necessary condition for a correct morph is then that the concatenation of the two deformations yield a mapping close to the identity.[1] Although this method can provide a partial quality criterion even when no ground truth is available, all model selection approaches based on such a criterion need to minimize (1) many times and the computation of a gradient with respect to the parameters is usually not possible. As the minimization is typically non-convex and rather expensive, the number of free parameters that can be optimized is small. For locally varying parameters as proposed here such an approach is not practical. We thus propose to learn the parameters from examples using an invariance criterion proposed in the next section.

## 4 Learning the optimal feature function

We assume that a database of $D$ objects that are already in correspondence is available. This could for example be achieved by manually picking many corresponding point pairs and training a regression to map all the points onto each other, or by (semi-)automatic methods optimized for the given object class (e.g., [1]). We can then determine the optimal approximation of the warp-invariant

feature function (as defined in the introduction) that characterizes correspondence using the basic features in our dictionary. The warp-invariant feature function should be such that it varies little or not at all for corresponding points, but its value should not be preserved (and have large variance) for random non-matching points. To approximate it, we propose to maximize the ratio of these variances over all weighting functions $\gamma$. Thus for each point $x \in \mathcal{X}$, we maximize

$$\frac{\mathbb{E}_{d,z_d} \left( f_\gamma^r(x) - f_\gamma^d(z_d) \right)^2}{\mathbb{E}_d \left( f_\gamma^r(x) - f_\gamma^d(\tau_d(x)) \right)^2} \tag{2}$$

Here, $f_\gamma^r, f_\gamma^d$ are the warp-invariant feature functions evaluated on the reference object and the $d$-th database object respectively. $\tau_d(x)$ is the point matching $x$ on the $d$-th database object and $z_d$ is a random point sampled from it. We take the expectations over all objects in our database, as well as non corresponding points randomly sampled from the objects.

Because of the linear dependence of $f_\gamma$ on $\gamma$ one can rewrite the problem as the maximization of

$$\frac{\gamma(x)^T C_z(x)\gamma(x)}{\gamma(x)^T C_\tau(x)\gamma(x)} \tag{3}$$

with the empirical covariances

$$[C_\tau(x)]_{i,j} = \sum_{d=1}^{D} \left( f_i^r(x) - f_i^d(\tau_d(x)) \right) \left( f_j^r(x) - f_j^d(\tau_d(x)) \right)^T, \tag{4}$$

$$[C_z(x)]_{i,j} = \sum_{d=1}^{D} \sum_{k=1}^{N} \left( f_i^r(x) - f_i^d(z_{d,k}) \right) \left( f_j^r(x) - f_j^d(z_{d,k}) \right)^T, \tag{5}$$

where we have drawn $N$ random sample points from each object in the database.

This problem is known as *oriented PCA* [13], and the maximizing vector $\gamma(x)$ can be determined by solving the generalized eigenvalue problem $C_\tau(x)v(x) = \lambda(x)C_z(x)v(x)$. If $v(x)$ is the normalized eigenvector corresponding to the maximal eigenvalue $\lambda(x)$, we obtain the optimal weight vector $\gamma(x) = \tilde{\lambda}(x)v(x)$ using the scale factor $\tilde{\lambda}(x) = \left( v(x)^T C_\tau(x)v(x) \right)^{-1/2}$.

Note that by using this scale factor $\tilde{\lambda}(x)$, the contribution of the feature function $f_\gamma$ in the objective (1) will vary locally compared to the regularizer: as $\tau(x)$ is somewhat arbitrary during the optimization of (1) the average local contribution will then approximately equal $\mathbb{E}_{d,z_d} \left( f_\gamma^r(x) - f_\gamma^d(z_d) \right)^2 = \lambda(x)$. This implies that if locally there exists a characteristic combination of features — $\lambda(x)$ is high — it will have a big influence in (1). If not, the smoothness term $\|\tau\|_{\mathcal{H}}$ gets relatively more weight implying that the local correspondence is mostly determined through more global contributions.

Note, moreover that while we have described the above for the leading eigenvector only, nothing prevents us from computing several eigenvectors and stacking up the resulting warp-invariant feature functions $f_\gamma^1, f_\gamma^2, \ldots, f_\gamma^m$ into a vector valued warp-invariant feature function $f_\gamma : \mathcal{X} \to \mathbb{R}^m$ which then is plugged into the optimization problem (1) using the two norm to measure deviations instead of the squared distance.

## 5    Basic Feature Functions

In our dictionary of basic feature functions we included the signed distance function and its derivative. We added a curvature related feature, the "signed balls", and surface texture intensity.

### 5.1    Signed Balls

Imagine a point $x$ on a flat piece of a surface. Take a ball $B_R(x)$ with radius $R$ centered at that point and compute the average of the signed distance function $s : \mathcal{X} \to \mathbb{R}$ over the ball's volume:

$$I_s(x) = \frac{1}{V_{B_R(x)}} \int_{B_R(x)} s(x')dx' - s(x) \tag{6}$$

If the surface around $x$ is flat on the scale of the ball, we obtain zero. At points where the surface is bent outwards this value is positive, at concave points it is negative. The normalization to the value

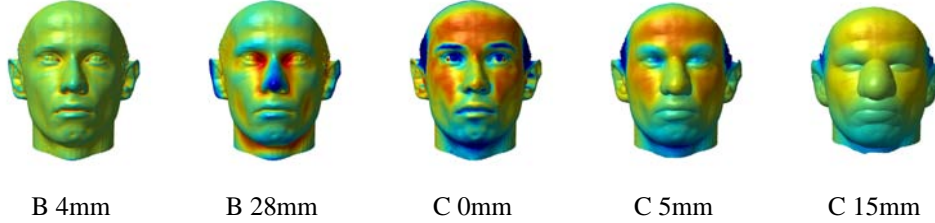

| B 4mm | B 28mm | C 0mm | C 5mm | C 15mm |

Figure 1: *The two figures on the left show the color-coded values of the "signed balls" feature at different radii $R$. Depending on $R$, the feature is sensitive to small-scale structures or large-scale structures only. Convex parts of the surface are assigned positive values (blue), concave parts negative (red). The three figures on the right show how the surface feature function that was trained with texture intensity extends off the surface (for clarity visualized in false colors) and becomes smoother. In the figure, the function is mapped on surfaces that are offset by 0, 5 and 15 mm.*

of the signed distance function at the center of the ball allows us to compute this feature function also for off-surface points, where the interpretation with respect to the other iso-surfaces does not change. Due to the integration, this feature is stable with respect to surface noise, while mean curvature in differential geometry may be affected significantly. Moreover, the integration involves a scale of the feature.

We propose to represent the implicit surface function as in [10] where a compactly supported kernel expansion is trained to approximate the signed distance. In this case the integral and the kernel summation can be interchanged, so we only need to evaluate terms of the form $\int_{B_R(x)} k(x_i, x') dx'$ and then add them in the same way as the signed distance function is computed. The value of this basic integral only depends on the distance between the kernel center $x_i$ and the test point $x$. It is compactly supported if the kernel $k$ is. Therefore, we propose to pre-compute these values numerically for different distances and store them in a small lookup table. For the final expansion summation we can then just interpolate the two closest values. We obtained good interpolation results with about ten to twenty distance values.

For the case where the surface looks locally like a sphere it is easy to show that in the limit of small balls the value of the "signed balls" feature function is related to the differential geometric mean curvature $H$ by $I_s(x) = \frac{3\pi}{20} H^2 R^2 + O(R^3)$.

## 5.2 Surface properties — Texture

The volume deformation approach presented in Section 3 requires the use of feature functions defined on the whole domain $\mathcal{X}$. In order to include information $f|_{\partial\Omega}$ which is just given on a surface $\partial\Omega$ of the object whose interior volume is $\Omega$, e.g. the texture intensity, we propose to extended the surface feature $f|_{\partial\Omega}$ into a differentiable feature function $f : \mathcal{X} \to \mathbb{R}$ such that $f \to f|_{\partial\Omega}$ as we get closer to the surface. At larger distances from the surface, $f$ should be smoother and tend towards the mean feature value. This is a desirable property during the optimization of (1) as it helps to avoid local minima. Finally, the feature function $f$ and its gradient should be efficient to evaluate.

We propose to use a multi-scale compactly supported kernel regression to determine $f$: at each scale, from coarse to fine, we select approximately equally spaced points on the surface at a distance related to the kernel width of that scale. Then we compute the feature value at these points averaged over a sphere of radius of the corresponding kernel support. With standard quadratic SVR regression we fit the remainder of what was achieved on larger scales to the training values. Due to the sub-sampling the kernel regressions do not contain too many kernel centers and the compact support of the kernel ensures sparse kernel matrices. Thus, efficient regression and evaluation is guaranteed. Because all kernel centers lie on the surface and reach to different extents into the volume $\mathcal{X}$ depending on the kernel size of their scale, we can model small-scale variations on the surface and close to it, whereas the regression function varies only on a larger scale further away from the surface.

## 6   Experiments

**Implementation.**   In order to optimize (1) we followed the approach of [10]: we represent the deformation $\tau$ as a multi-scale compactly supported kernel expansion, i.e., the $j$-th component,

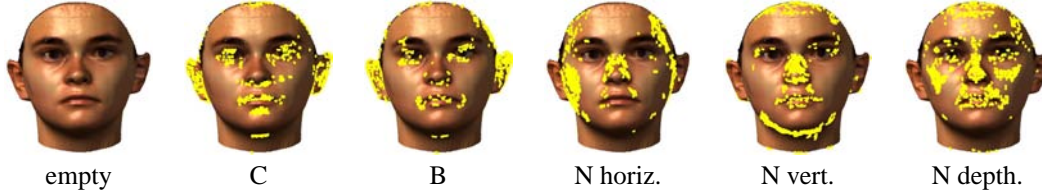

| empty | C | B | N horiz. | N vert. | N depth. |

Figure 2: *Locations that are marked yellow show an above threshold, relative contribution (see text) of a given feature in the warp-invariant feature function. C is the surface intensity feature, B the signed balls feature ($R = 6mm$), N the surface normals in different directions. Note that points where color has a large contribution (yellow points in C) are clustered around regions with characteristic color information, such as the eyes or the mouth.*

$j = 1, 2, 3$, of $\tau$ is $\tau^j(x) = x^j + \sum_{s=1}^{S} \sum_{i=1}^{N_s} \alpha_{i,s}^j k(x, x_{i,s})$ with the compactly supported kernel function $k : \mathcal{X} \times \mathcal{X} \to \mathbb{R}$. The regularizer then is $\|\tau\|_{\mathcal{H}}^2 := \sum_{s=1}^{S} \sum_{j=1}^{3} \sum_{i,l=1}^{N_s} \alpha_{i,s}^j \alpha_{l,s}^j k(x_{l,s}, x_{i,s})$. We approximate the integral in (1) by sampling $N_s$ kernel centers $x_{i,s}$ on each scale $s = 1, \ldots, S$ according to the measure $\mu(x)$ and minimize the resulting non-linear optimization problem in the coefficients $\alpha_{i,s}^j$ for each scale from coarse to fine using a second order Newton-like method [14].

As a test object class we used 3D heads with known correspondence [1]. 100 heads were used for the training object database and 10 to test our correspondence algorithm. As a reference head we used the mean head of the database. The faces are all in correspondence, so we can just linearly average the vertex positions and the texture images. However, the correspondence of the objects in the database is only defined on the surface. In order to extend it to the off-surface points $x_{i,s}$, we generated these locations by first sampling points from the surface and then displacing them along their surface normals. This implied that we were able to identify the corresponding points also on other heads.

For each kernel center $x_{i,s}$, we learned the weighting vector $\gamma(x_{i,s})$ as described in Section 4. In one run through the database we computed for each head the values of all proposed basic feature functions for all locations, corresponding to kernel centers on the reference head, as well as for 100 randomly sampled points $z$. The points $z$ should be typical for possible target locations $\tau(x_{i,s})$ during the optimization of (1). Thus, we sampled points up to distances to the surface proportional to the kernel widths used for the deformation $\tau$. We then estimated the empirical covariance matrices for each kernel center yielding the weight vectors via a small eigenvalue decomposition of size $n \times n$ where $n$ is the number of used basic features. The parameters $C_{reg}$ — one for each scale — were determined by optimizing computed deformation fields from the reference head to some of the training database heads. We minimized the mismatch to the correspondence given in the database.

**Feature functions.** In Figure 1, our new feature functions are visualized on an example head. Each feature extracts specific plausible information, and the surface color can be extended off the surface.

**Learned weights.** In Figure 2, we have marked those points on the surface where a given feature has a high relative contribution in the warp-invariant feature function. As a measure of contribution we took the component of the weight vector $\gamma(x_{i,s})$ that corresponds to the feature of interest and multiplied it with the standard deviation of this feature over all heads and all positions. Note that the weight vector is not invariant to rescaling the basic feature functions, unlike the proposed measure. Finally, we normalized the contributions of all features at a given point $x_{i,s}$ to sum to one, yielding the relative contribution. In the table below the relative contribution of each feature is listed.

|  | S | N horiz. | N vert. | N depth. | C | B 8% | B 3% |
|---|---|---|---|---|---|---|---|
| average rel. contribution | 0.832 | 0.092 | 0.023 | 0.038 | 0.008 | 0.006 | 0.003 |
| max rel. contribution | 0.997 | 0.701 | 0.429 | 0.446 | 0.394 | 0.272 | 0.333 |

Here and below, S is signed distance, N surface normals, C the proposed surface feature function trained with the intensity values on the faces, and B is the "signed balls" feature with radii given by the percentage numbers scaled to the diameter of the head.

The signed distance function is the best preserved feature (e.g. all surface points take the value zero up to small approximation errors). The resulting large weight of this feature is plausible as a surface-to-surface mapping is a necessary condition for a morph. However, combined with Figure 2

| | Reference | Deformed | Target | Deformed | Target |
|---|---|---|---|---|---|

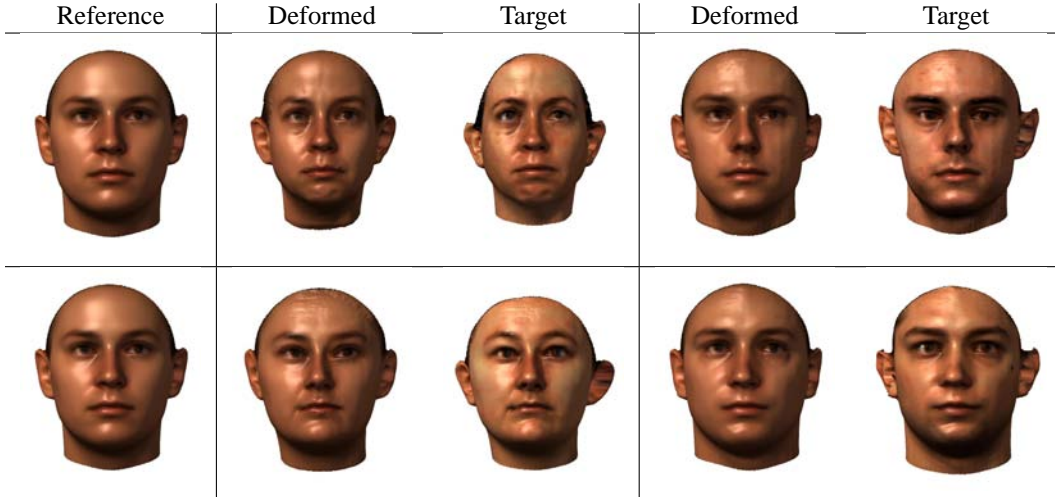

Figure 3: *The average head of the database – the reference – is deformed to match four of the target heads of the test set. Correct correspondence deforms the shape of the reference head to the target face with the texture of the mean face well aligned to the shape details.*

we can see that the method assigns plausible non-zero values also to other features where these can be assumed to be most characteristic for a good correspondence.

**Correspondence.** We applied our correspondence algorithm to compute the correspondence to the test set of 10 heads. Some example deformations are shown in Figure 3 for a dictionary consisting of S, N (hor, vert, depth) C, B (radii 3% and 8%). Numerical evaluation of the morphs is difficult. We compare our method with the results of the correspondence algorithm of [1] on points that are uniformly drawn form the surface (first column) and for 24 selected marker points (second column). These markers were placed at locations around the eyes or the mouth where correspondence can be assumed to be better defined than for example on the forehead. Still, the error made by humans when picking these positions has turned out to be around 1–2mm. The table below shows mean results in mm for different settings.

| | | uniform | markers | error signed distance |
|---|---|---|---|---|
| (a) | all weights equal | 5.97 | 4.49 | 1.49 |
| (b) | our method (independent of $x$) | 3.74 | 1.48 | 0.05 |
| (c) | our method (1 eigenvector) | 3.74 | 1.34 | 0.04 |
| (d) | our method (2 eigenvectors) | 3.62 | 1.19 | 0.04 |
| (e) | our method (4 eigenvectors) | 3.56 | 1.11 | 0.04 |
| (f) | our method (6 eigenvectors) | 3.55 | 1.10 | 0.04 |
| (g) | our method (1 eigenvector, without B, C) | 3.76 | 1.42 | 0.04 |

If all weights are equal independent of location or feature (a), the result is not acceptable. A careful weighting of each feature separately, but independent of location (b) — as could potentially be achieved by [10] — improves the quality of the correspondence. To obtain these weights we averaged the covariance matrices $C_z(x), C_\tau(x)$ over all points and applied the proposed algorithm in Section 4, but independent of $x$. However, a locally adapted weighting (c) outperforms the above methods and using more than one eigenvector (d-f) further enhances the correspondence. Note that although the results are not identical to [1], our algorithm's accuracy is consistent with the human labeling on the scale of the latter's presumed accuracy (1-2mm). For uniformly sampled points, the differences are slightly larger (4mm), but we need to bear in mind that that algorithm's results cannot be considered ground truth. Experiment (g) which is identical to (c) but with the color and signed balls feature omitted demonstrates the usefulness of these additional basic feature functions.

Computation times ranged between 5min and one hour and depended significantly on the number of scales used (here 4), the number of kernel centers generated, and the number of basic features included in the dictionary. For large radii $R$ the signed balls feature becomes quite expensive to compute, since many summands of the signed distance function expansion have to be accumulated. Our method to select the important features for each point in advance, i.e. before the optimization is

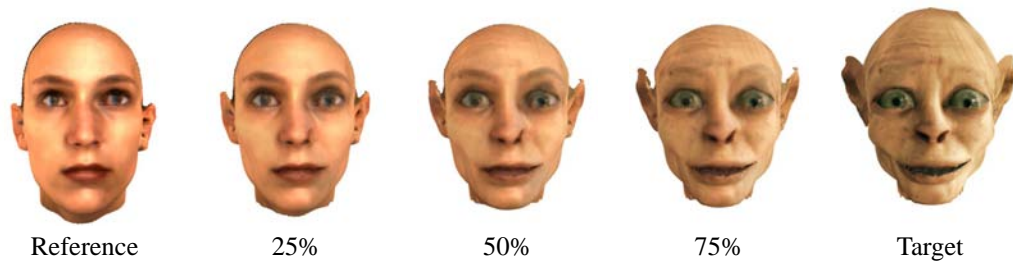

| Reference | 25% | 50% | 75% | Target |

Figure 4: *A morph between a human head and the head of the character Gollum (available from* **www.turbosquid.com***). As Gollum's head falls out of our object class (human heads), we assisted the training procedure with 28 manually placed markers.*

started, would allow for a potentially high speed-up: At locations where a certain feature has a very low weight, we could just omit it in the evaluation of the cost function (1).

## 7 Conclusion

We have proposed a new approach to the challenging problem of defining criteria that characterize a valid correspondence between 3D objects of a given class. Our method learns an appropriate criterion from examples of correct correspondences. The approach thus applies machine learning to computer graphics at the early level of feature construction. The learning technique has been implemented efficiently in a correspondence algorithm for textured surfaces.

In the future, we plan to test our method with other object classes. Even though we have concentrated in our experiments on 3D surface data, the method may be applicable also in other fields such as to align CT or MR scans in medical imaging. It would also be intriguing to explore the question whether our paradigm of learning the features characterizing correspondences might reflect some of the cognitive processes that are involved when humans learn about similarities within object classes.

## Footnotes

[1]It is not a sufficient condition, since the concatenation of, say, two identity mappings will also yield the identity.

## References

[1] V. Blanz and T. Vetter. A morphable model for the synthesis of 3d faces. In *SIGGRAPH'99 Conference Proceedings*, pages 187–194, Los Angeles, 1999. ACM Press.

[2] B.D. Lucas and T. Kanade. An iterative image registration technique with an application to stereo vision. In *IJCAI81*, pages 674–679, 1981.

[3] B. K. P. Horn and B. G. Schunck. Determining optical flow. *Artif. Intell.*, 17(1-3):185–203, 1981.

[4] D. Beymer and T. Poggio. Image representations for visual learning. *Science*, 272:1905–1909, 1996.

[5] T. Vetter and T. Poggio. Linear object classes and image synthesis from a single example image. *IEEE Trans. on Pattern Analysis and Machine Intelligence*, 19(7):733–742, 1997.

[6] B. Allen, B. Curless, and Z. Popovic. The space of human body shapes: reconstruction and parameterization from range scans. In *Proc. SIGGRAPH*, pages 612–619, 2002.

[7] D. Rueckert and A. F. Frangi. Automatic construction of 3-d statistical deformation models of the brain using nonrigid registration. *IEEE Trans. on Medical Imaging*, 22(8):1014–1025, 2003.

[8] D. Anguelov, P. Srinivasan, H.-C. Pang, D. Koller, S. Thrun, and J. Davis. The correlated correspondence algorithm for unsupervised registration of nonrigid surfaces. In *Neural Information Processing Systems 17*, pages 33–40. MIT Press, 2005.

[9] M. Alexa. Recent advances in mesh morphing. *Computer Graphics Forum*, 21(2):173–196, 2002.

[10] B. Schölkopf, F. Steinke, and V. Blanz. Object correspondence as a machine learning problem. In *Proceedings of the 22nd International Conference on Machine Learning (ICML 05)*, July 2005.

[11] J.-P. Thirion and A Gourdon. Computing the differential characteristics of isointensity surfaces. *Journal of Computer Vision and Image Understanding*, 61(2):190–202, March 1995.

[12] N. Gelfand, N. J. Mitra, L. J. Guibas, and H. Pottmann. Robust global registration. In *Proc. Eurographics Symposium on Geometry Processing*, pages 197–206, 2005.

[13] K.I. Diamantaras and S.Y. Kung. *Principal component neural networks: theory and applications*. John Wiley & Sons, Inc., 1996.

[14] D. C. Liu and J. Nocedal. On the limited memory bfgs method for large scale optimization. *Math. Program.*, 45(3):503–528, 1989.
